# Reinforcement Learning for Dynamic Channel Allocation in Cellular Telephone Systems

**Satinder Singh**
Department of Computer Science
University of Colorado
Boulder, CO 80309-0430
baveja@cs.colorado.edu

**Dimitri Bertsekas**
Lab. for Info. and Decision Sciences
MIT
Cambridge, MA 02139
bertsekas@lids.mit.edu

## Abstract

In cellular telephone systems, an important problem is to dynamically allocate the communication resource (channels) so as to maximize service in a stochastic caller environment. This problem is naturally formulated as a dynamic programming problem and we use a reinforcement learning (RL) method to find dynamic channel allocation policies that are better than previous heuristic solutions. The policies obtained perform well for a broad variety of call traffic patterns. We present results on a large cellular system with approximately $49^{49}$ states.

In cellular communication systems, an important problem is to allocate the communication resource (bandwidth) so as to maximize the service provided to a set of mobile callers whose demand for service changes stochastically. A given geographical area is divided into mutually disjoint cells, and each cell serves the calls that are within its boundaries (see Figure 1a). The total system bandwidth is divided into channels, with each channel centered around a frequency. Each channel can be used simultaneously at different cells, provided these cells are sufficiently separated spatially, so that there is no interference between them. The minimum separation distance between simultaneous reuse of the same channel is called the *channel reuse constraint*.

When a call requests service in a given cell either a free channel (one that does not violate the channel reuse constraint) may be assigned to the call, or else the call is blocked from the system; this will happen if no free channel can be found. Also, when a mobile caller crosses from one cell to another, the call is "handed off" to the cell of entry; that is, a new free channel is provided to the call at the new cell. If no such channel is available, the call must be dropped/disconnected from the system.

One objective of a channel allocation policy is to allocate the available channels to calls so that the number of blocked calls is minimized. An additional objective is to minimize the number of calls that are dropped when they are handed off to a busy cell. These two objectives must be weighted appropriately to reflect their relative importance, since dropping existing calls is generally more undesirable than blocking new calls.

To illustrate the qualitative nature of the channel assignment decisions, suppose that there are only two channels and three cells arranged in a line. Assume a channel reuse constraint of 2, i.e., a channel may be used simultaneously in cells 1 and 3, but may not be used in channel 2 if it is already used in cell 1 or in cell 3. Suppose that the system is serving one call in cell 1 and another call in cell 3. Then serving both calls on the same channel results in a better channel usage pattern than serving them on different channels, since in the former case the other channel is free to be used in cell 2. The purpose of the channel assignment and channel rearrangement strategy is, roughly speaking, to create such favorable usage patterns that minimize the likelihood of calls being blocked.

We formulate the channel assignment problem as a dynamic programming problem, which, however, is too complex to be solved exactly. We introduce approximations based on the methodology of reinforcement learning (RL) (e.g., Barto, Bradtke and Singh, 1995, or the recent textbook by Bertsekas and Tsitsiklis, 1996). Our method learns channel allocation policies that outperform not only the most commonly used policy in cellular systems, but also the best heuristic policy we could find in the literature.

# 1   CHANNEL ASSIGNMENT POLICIES

Many cellular systems are based on a *fixed assignment* (FA) channel allocation; that is, the set of channels is partitioned, and the partitions are permanently assigned to cells so that all cells can use all the channels assigned to them simultaneously without interference (see Figure 1a). When a call arrives in a cell, if any pre-assigned channel is unused; it is assigned, else the call is blocked. No rearrangement is done when a call terminates. Such a policy is static and cannot take advantage of temporary stochastic variations in demand for service. More efficient are *dynamic channel allocation* policies, which assign channels to different cells, so that every channel is available to every cell on a need basis, unless the channel is used in a nearby cell and the reuse constraint is violated.

The best existing dynamic channel allocation policy we found in the literature is Borrowing with Directional Channel Locking (BDCL) of Zhang & Yum (1989). It numbers the channels from 1 to $N$, partitions and assigns them to cells as in FA. The channels assigned to a cell are its nominal channels. If a nominal channel is available when a call arrives in a cell, the smallest numbered such channel is assigned to the call. If no nominal channel is available, then the largest numbered free channel is borrowed from the neighbour with the most free channels. When a channel is borrowed, careful accounting of the directional effect of which cells can no longer use that channel because of interference is done. The call is blocked if there are no free channels at all. When a call terminates in a cell and the channel so freed is a nominal channel, say numbered $i$, of that cell, then if there is a call in that cell on a borrowed channel, the call on the smallest numbered borrowed channel is reassigned to $i$ and the borrowed channel is returned to the appropriate cell. If there is no call on a borrowed channel, then if there is a call on a nominal channel numbered larger than $i$, the call on the highest numbered nominal channel is reassigned to $i$. If the call just terminated was itself on a borrowed channel, the

call on the smallest numbered borrowed channel is reassigned to it and that channel is returned to the cell from which it was borrowed. Notice that when a borrowed channel is returned to its original cell, a nominal channel becomes free in that cell and triggers a reassignment. Thus, in the worst case a call termination in one cell can sequentially cause reassignments in arbitrarily far away cells — making BDCL somewhat impractical.

BDCL is quite sophisticated and combines the notions of channel-ordering, nominal channels, and channel borrowing. Zhang and Yum (1989) show that BDCL is superior to its competitors, including FA. Generally, BDCL has continued to be highly regarded in the literature as a powerful heuristic (Enrico et.al., 1996). In this paper, we compare the performance of dynamic channel allocation policies learned by RL with both FA and BDCL.

## 1.1   DYNAMIC PROGRAMMING FORMULATION

We can formulate the dynamic channel allocation problem using dynamic programming (e.g., Bertsekas, 1995). State transitions occur when channels become free due to call departures, or when a call arrives at a given cell and wishes to be assigned a channel, or when there is a handoff, which can be viewed as a simultaneous call departure from one cell and a call arrival at another cell. The state at each time consists of two components:

(1) The list of occupied and unoccupied channels at each cell. We call this the configuration of the cellular system. It is *exponential* in the number of cells.

(2) The event that causes the state transition (arrival, departure, or handoff). This component of the state is uncontrollable.

The decision/control applied at the time of a call departure is the rearrangement of the channels in use with the aim of creating a more favorable channel packing pattern among the cells (one that will leave more channels free for future assignments). Unlike BDCL, our RL solution will restrict this rearrangement to the cell with the current call departure. The control exercised at the time of a call arrival is the assignment of a free channel, or the blocking of the call if no free channel is currently available. In general, it may also be useful to do *admission control*, i.e., to allow the possibility of not accepting a new call even when there exists a free channel to minimize the dropping of ongoing calls during handoff in the future. We address admission control in a separate paper and here restrict ourselves to always accepting a call if a free channel is available. The objective is to learn a policy that assigns decisions (assignment or rearrangement depending on event) to each state so as to *maximize*

$$J = E\left\{\int_0^\infty e^{-\beta t} c(t) dt\right\},$$

where $E\{\cdot\}$ is the expectation operator, $c(t)$ is the number of ongoing calls at time $t$, and $\beta$ is a discount factor that makes immediate profit more valuable than future profit. Maximizing $J$ is equivalent to minimizing the expected (discounted) number of blocked calls over an infinite horizon.

## 2   REINFORCEMENT LEARNING SOLUTION

RL methods solve optimal control (or dynamic programming) problems by learning good approximations to the optimal value function, $J^*$, given by the solution to

the Bellman optimality equation which takes the following form for the dynamic channel allocation problem:

$$J(x) \;=\; E_e \left\{ \max_{a \in A(x,e)} [E_{\Delta t}\{c(x, a, \Delta t) + \gamma(\Delta t)J(y)\}] \right\}, \qquad (1)$$

where $x$ is a configuration, $e$ is the random event (a call arrival or departure), $A(x, e)$ is the set of actions available in the current state $(x, e)$, $\Delta t$ is the random time until the next event, $c(x, a, \Delta t)$ is the effective immediate payoff with the discounting, and $\gamma(\Delta t)$ is the effective discount for the next configuration $y$.

RL learns approximations to $J^*$ using Sutton's (1988) temporal difference (TD(0)) algorithm. A fixed feature extractor is used to form an approximate compact representation of the exponential configuration of the cellular array. This approximate representation forms the input to a function approximator (see Figure 1) that learns/stores estimates of $J^*$. No partitioning of channels is done; all channels are available in each cell. On each event, the estimates of $J^*$ are used both to make decisions and to update the estimates themselves as follows:

**Call Arrival**: When a call arrives, evaluate the next configuration for each free channel and assign the channel that leads to the configuration with the largest estimated value. If there is no free channel at all, no decision has to be made.

**Call Termination**: When a call terminates, one by one each ongoing call in that cell is considered for reassignment to the just freed channel; the resulting configurations are evaluated and compared to the value of not doing any reassignment at all. The action that leads to the highest value configuration is then executed.

On call arrival, as long as there is a free channel, the number of ongoing calls and the time to next event do not depend on the free channel assigned. Similarly, the number of ongoing calls and the time to next event do not depend on the rearrangement done on call departure. Therefore, both the sample immediate payoff which depends on the number of ongoing calls and the time to next event, and the effective discount factor which depends only on the time to next event are independent of the choice of action. Thus one can choose the current best action by simply considering the estimated values of the next configurations. The next configuration for each action is deterministic and trivial to compute.

When the next random event occurs, the sample payoff and the discount factor become available and are used to update the value function as follows: on a transition from configuration $x$ to $y$ on action $a$ in time $\Delta t$,

$$J_{new}(\tilde{x}) \;=\; (1 - \alpha)J_{old}(\tilde{x}) + \alpha \left(c(x, a, \Delta t) + \gamma(\Delta t)J_{old}(\tilde{y})\right) \qquad (2)$$

where $\tilde{x}$ is used to indicate the approximate feature-based representation of $x$. The parameters of the function approximator are then updated to best represent $J_{new}(\tilde{x})$ using gradient descent in mean-squared error $(J_{new}(\tilde{x}) - J_{old}(\tilde{x}))^2$.

## 3  SIMULATION RESULTS

Call arrivals are modeled as Poisson processes with a separate mean for each cell, and call durations are modeled with an exponential distribution. The first set of results are on the 7 by 7 cellular array of Figure ??a with 70 channels (roughly $70^{49}$ configurations) and a channel reuse constraint of 3 (this problem is borrowed from Zhang and Yum's (1989) paper on an empirical comparison of BDCL and its competitors). Figures 2a, b & c are for uniform call arrival rates of 150, 200, and 350 calls/hr respectively in each cell. The mean call duration for all the experiments

reported here is 3 minutes. Figure 2d is for non-uniform call arrival rates. Each curve plots the cumulative empirical blocking probability as a function of simulated time. Each data point is therefore the percentage of system-wide calls that were blocked up until that point in time. All simulations start with no ongoing calls.

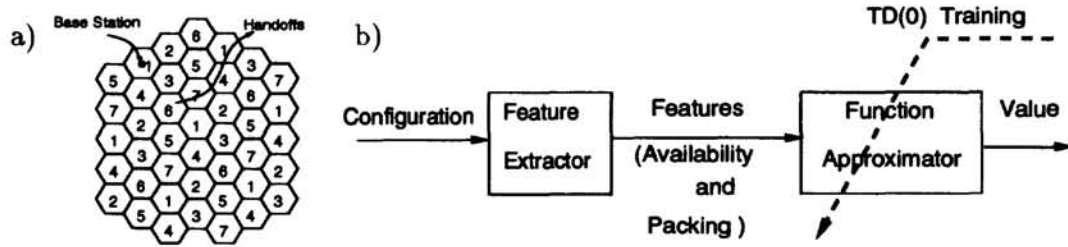

Figure 1: a) Cellular Array. The market area is divided up into cells, shown here as hexagons. The available bandwidth is divided into channels. Each cell has a base station responsible for calls within its area. Calls arrive randomly, have random durations and callers may move around in the market area creating handoffs. The channel reuse constraint requires that there be a minimum distance between simultaneous reuse of the same channel. In a fixed assignment channel allocation policy (assuming a channel reuse constraint of 3), the channels are partitioned into 7 lots labeled 1 to 7 and assigned to the cells in the compact pattern shown here. Note that the minimum distance between cells with the same number is at least three. b) A block diagram of the RL system. The exponential configuration is mapped into a feature-based approximate representation which forms the input to a function approximation system that learns values for configurations. The parameters of the function approximator are trained using gradient descent on the squared TD(0) error in value function estimates (c.f. Equation 2).

The RL system uses a linear neural network and two sets of features as input: one availability feature for each cell and one packing feature for each cell-channel pair. The availability feature for a cell is the number of free channels in that cell, while the packing feature for a cell-channel pair is the number of times that channel is used in a 4 cell radius. Other packing features were tried but are not reported because they were insignificant. The RL curves in Figure 2 show the empirical blocking probability whilst learning. Note that learning is quite rapid. As the mean call arrival rate is increased the relative difference between the 3 algorithms decreases. In fact, FA can be shown to be optimal in the limit of infinite call arrival rates (see McEliece and Sivarajan, 1994). With so many customers in every cell there are no short-term fluctuations to exploit. However, as demonstrated in Figure 2, for practical traffic rates RL consistently gives a big win over FA and a smaller win over BDCL.

One difference between RL and BDCL is that while the BDCL policy is independent of call traffic, RL adapts its policy to the particulars of the call traffic it is trained on and should therefore be less sensitive to different patterns of non-uniformity of call traffic across cells. Figure 3b presents multiple sets of bar-graphs of asymptotic blocking probabilities for the three algorithms on a 20 by 1 cellular array with 24 channels and a channel reuse constraint of 3. For each set, the average per-cell call arrival rate is the same (120 calls/hr; mean duration of 3 minutes), but the pattern of call arrival rates across the 20 cells is varied. The patterns are shown on the left of the bar-graphs and are explained in the caption of Figure 3b. From Figure 3b it is apparent that RL is much less sensitive to varied patterns of non-uniformity than both BDCL and FA.

We have showed that RL with a linear function approximator is able to find better dynamic channel allocation policies than the BDCL and FA policies. Using nonlinear neural networks as function approximators for RL did in some cases improve

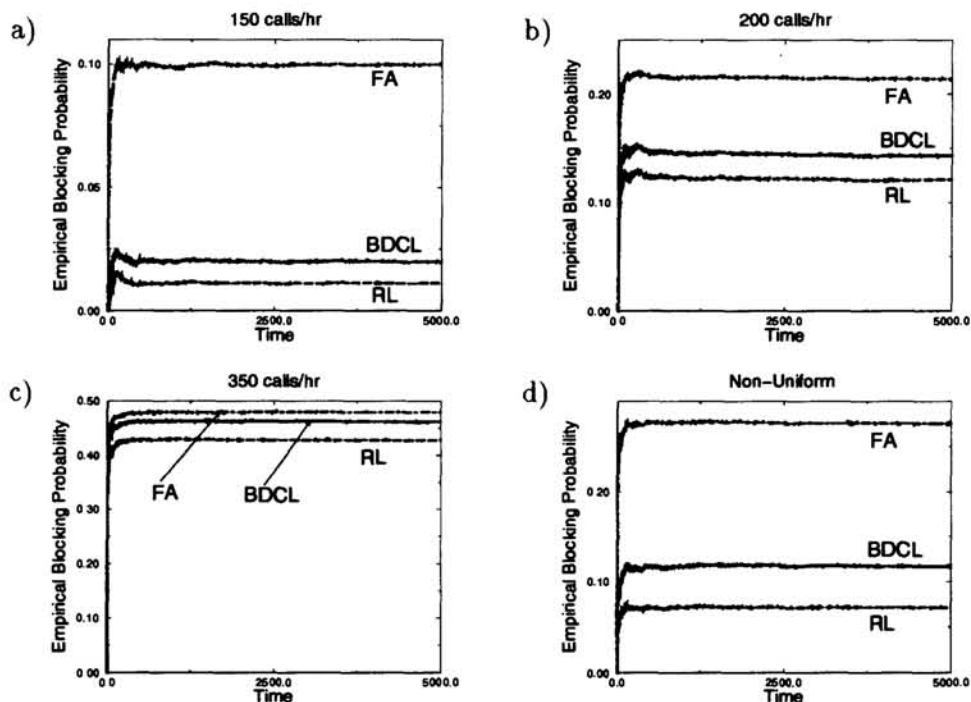

Figure 2: a), b), c) & d) These figures compare performance of RL, FA, and BDCL on the 7 by 7 cellular array of Figure 1a. The means of the call arrival (Poisson) processes are shown in the graph titles. Each curve presents the cumulative empirical blocking probability as a function of time elapsed in minutes. All simulations start with no ongoing calls and therefore the blocking probabilities are low in the early minutes of the performance curves. The RL curves presented here are for a linear function approximator and show performance while learning. Note that learning is quite rapid.

performance over linear networks by a small amount but at the cost of a big increase in training time. We chose to present results for linear networks because they have the advantage that even though training is centralized, the policy so learned is decentralized because the features are local and therefore just the weights from the local features in the trained linear network can be used to choose actions in each cell. For large cellular arrays, training itself could be decentralized because the choice of action in a particular cell has a minor effect on far away cells. We will explore the effect of decentralized training in future work.

## 4   CONCLUSION

The dynamic channel allocation problem is naturally formulated as an optimal control or dynamic programming problem, albeit one with very large state spaces. Traditional dynamic programming techniques are computationally infeasible for such large-scale problems. Therefore, knowledge-intensive heuristic solutions that ignore the optimal control framework have been developed. Recent approximations to dynamic programming introduced in the reinforcement learning (RL) community make it possible to go back to the channel assignment problem and solve it as an optimal control problem, in the process finding better solutions than previously available. We presented such a solution using Sutton's (1988) TD(0) with a feature-based linear network and demonstrated its superiority on a problem with approximately $70^{49}$ states. Other recent examples of similar successes are the game

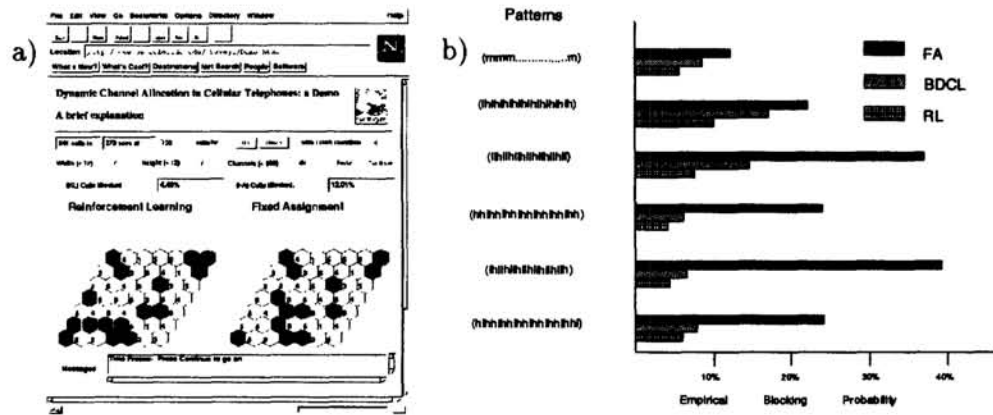

Figure 3: a) Screen dump of a Java Demonstration available publicly at http://www.cs.colorado.edu/~baveja/Demo.html b) Sensitivity of channel assignment methods to non-uniform traffic patterns. This figure plots asymptotic empirical blocking probability for RL, BDCL, and FA for a linear array of cells with different patterns (shown at left) of mean call arrival rates — chosen so that the average per cell call arrival rate is the same across patterns. The symbol $l$ is for low, $m$ for medium, and $h$ for high. The numeric values of $l$, $h$, and $m$ are chosen separately for each pattern to ensure that the average per cell rate of arrival is 120 calls/hr. The results show that RL is able to adapt its allocation strategy and thereby is better able to exploit the non-uniform call arrival rates.

of backgammon (Tesauro, 1992), elevator-scheduling (Crites & Barto, 1995), and job-shop scheduling (Zhang & Dietterich, 1995). The neuro-dynamic programming textbook (Bertsekas and Tsitsiklis, 1996) presents a variety of related case studies.

## References

Barto, A.G., Bradtke, S.J. & Singh, S. (1995) Learning to act using real-time dynamic programming. *Artificial Intelligence*, 72:81–138.

Bertsekas, D.P. (1995) *Dynamic Programming and Optimal Control: Vols 1 and 2.* Athena-Scientific, Belmont, MA.

Bertsekas, D.P. & Tsitsiklis, J. (1996) *Neuro-Dynamic Programming* Athena-Scientific, Belmont, MA.

Crites, R.H. & Barto, A.G. (1996) Improving elevator performance using reinforcement learning. In *Advances is Neural Information Processing Systems 8.*

Del Re, W., Fantacci, R. & Ronga, L. (1996) A dynamic channel allocation technique based on Hopfield Neural Networks. *IEEE Transactions on Vehicular Technology*, 45:1.

McEliece, R.J. & Sivarajan, K.N. (1994), Performance limits for channelized cellular telephone systems. *IEEE Trans. Inform. Theory*, pp. 21–34, Jan.

Sutton, R.S. (1988) Learning to predict by the methods of temporal differences. *Machine Learning*, 3:9–44.

Tesauro, G.J. (1992) Practical issues in temporal difference learning. *Machine Learning*, 8(3/4):257–277.

Zhang, M. & Yum, T.P. (1989) Comparisons of Channel-Assignment Strategies in Cellular Mobile Telephone Systems. *IEEE Transactions on Vehicular Technology* Vol. 38, No. 4.

Zhang, W. & Dietterich, T.G. (1996) High-performance job-shop scheduling with a time-delay TD(lambda) network. In *Advances is Neural Information Processing Systems 8.*
